# Non–Linear Dimensionality Reduction

**David DeMers* & Garrison Cottrell[†]**
Dept. of Computer Science & Engr., 0114
Institute for Neural Computation
University of California, San Diego
9500 Gilman Dr.
La Jolla, CA, 92093-0114

## Abstract

A method for creating a non–linear encoder–decoder for multidimensional data with compact representations is presented. The commonly used technique of autoassociation is extended to allow non–linear representations, and an objective function which penalizes activations of individual hidden units is shown to result in minimum dimensional encodings with respect to allowable error in reconstruction.

## 1 INTRODUCTION

Reducing dimensionality of data with minimal information loss is important for feature extraction, compact coding and computational efficiency. The data can be tranformed into "good" representations for further processing, constraints among feature variables may be identified, and redundancy eliminated. Many algorithms are exponential in the dimensionality of the input, thus even reduction by a single dimension may provide valuable computational savings.

Autoassociating feedforward networks with one hidden layer have been shown to extract the principal components of the data (Baldi & Hornik, 1988). Such networks have been used to extract features and develop compact encodings of the data (Cottrell, Munro & Zipser, 1989). Principal Components Analysis projects the data into a linear subspace

[†]email: gary@cs.ucsd.edu

**Non-Linear**
**"Principal Components" Net**

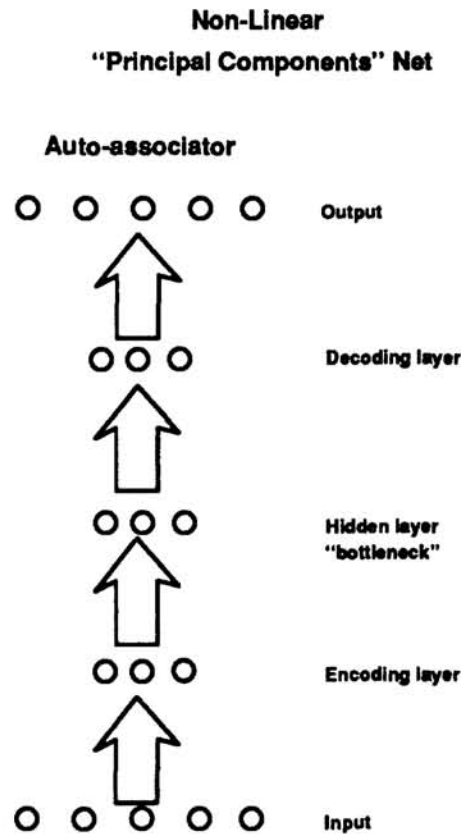

Figure 1: *A network capable of non–linear lower dimensional representations of data.*

with minimum information loss, by multiplying the data by the eigenvectors of the sample covariance matrix. By examining the magnitude of the corresponding eigenvalues one can estimate the minimum dimensionality of the space into which the data may be projected and estimate the loss. However, if the data lie on a non–linear submanifold of the feature space, then Principal Components will overestimate the dimensionality. For example, the covariance matrix of data sampled from a helix in $\mathbf{R}^3$ will have full–rank and thus three principal components. However, the helix is a one–dimensional manifold and can be (smoothly) parameterized with a single number.

The addition of hidden layers between the inputs and the representation layer, and between the representation layer and the outputs provides a network which is capable of learning non–linear representations (Kramer, 1991; Oja, 1991; Usui, Nakauchi & Nakano, 1991). Such networks can perform the non–linear analogue to Principal Components Analysis, and extract "principal manifolds". Figure 1 shows the basic structure of such a network. However, the dimensionality of the representation layer is problematic. Ideally, the dimensionality of the encoding (and hence the number of representation units needed) would be determined from the data.

We propose a pruning method for determining the dimensionality of the representation. A greedy algorithm which successively eliminates representation units by penalizing variances results in encodings of minimal dimensionality with respect to the allowable reconstruction error. The algorithm therefore performs non–linear dimensionality reduction (NLDR).

## 2   DIMENSIONALITY ESTIMATION BY REGULARIZATION

The *a priori* assignment of the number of units for the representation layer is problematic. In order to achieve maximum data compression, this number should be as small as possible; however, one also wants to preserve the information in the data and thus encode the data with minimum error. If the intrinsic dimensionality is not known ahead of time (as is typical), some method to estimate the dimensionality is desired. Minimization of the variance of a representation unit will essentially squeeze the variance of the data into the other hidden units. Repeated minimization results in increasingly lower–dimensional representation.

More formally, let the dimensionality of the raw data be $n$. We wish to find $F$ and its approximate inverse such that $\mathbf{R}^n \xrightarrow{F} \mathbf{R}^p \xrightarrow{F^{-1}} \mathbf{R}^n$ where $p < n$. Let $y$ denote the $p$–dimensional vector whose elements are the $p$ univalued functions $f_i$ which make up $F$. If one of the component functions $f_i$ is always constant, it is not contributing to the autoassociation and can be eliminated, yielding a function $F$ with $p - 1$ components. A constant value for $f_i$ means that the variance of $f_i$ over the data is zero. We add a regularization term to the objective function penalizing the variance of one of the representation units. If the variance can be driven to near zero while simultaneously achieving a target error in the primary task of autoassociation, then the unit being penalized can be pruned.

Let $H_p = \lambda_p (\Sigma_{j=1}^N (h_p(\text{net}^j) - E(h_p(\text{net}^j)))^2)$ where net$^j$ is the net input to the unit given the $j$th training pattern, $h_p(\text{net}^j)$ is the activation of the $p$th hidden unit in the representation layer (the one being penalized) and E is the expectation operator. For notational clarity, the superscripts will be suppressed hereafter. $E(h_i(x_j))$ can be estimated as $\bar{h}_p$, the mean activation of $h_p$ over all patterns in the training data.

$$\frac{\partial H_p}{\partial w_{pl}} = \frac{\partial H_p}{\partial \text{net}_p} \frac{\partial \text{net}_p}{\partial w_{pl}} = 2\lambda_p (h_p - \bar{h}_p) h_p' o_l$$

where $h_p'$ is the derivative of the activation function of unit $h_p$ with respect to its input, and $o_l$ is the output of the $l$th unit in the preceding layer. Let $\delta_p = 2\lambda_p h_p' (h_p - \bar{h}_p)$. We simply add $\delta_p$ to the delta of $h_p$ due to backpropagation from the output layer.

We first train a multi–layer[1] network to learn the identity map. When error is below a user–specified threshold, $\lambda_i$ is increased for the unit with lowest variance. If network weights can be found[2] such that the variance can be reduced below a small threshold while the remaining units are able to encode the data, the hidden unit in question is no longer contributing to the autoencoding, and its connections are excised from the network. The process is repeated until the variance of the unit in question cannot be reduced while maintaining low error.

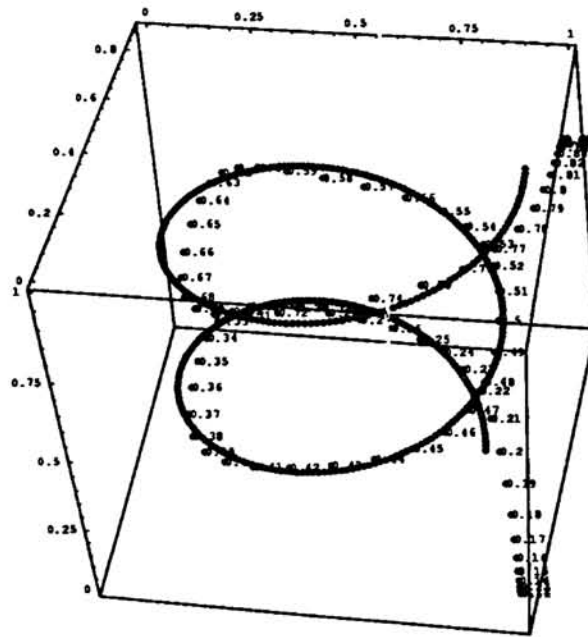

Figure 2: *The original 3–D helix data plus reconstruction from a single parameter encoding.*

# 3    RESULTS

We applied this method to several problems:

1. a closed 1–D manifold in $\mathbf{R}^3$.
2. a 1–D helix in $\mathbf{R}^3$.
3. Time series data generated from the Mackey–Glass delay–differential equation.
4. 160 64 by 64 pixel, 8-bit grayscale face images.

A number of parameter values must be chosen; error threshold, maximum magnitude of weights, value of $\lambda_i$ when increased, and when to "give up" training. For these experiments, they were chosen by hand; however, reasonable values can be selected such that the method can be automated.

## 3.1    Static Mappings: Circle and Helix

The first problem is interesting because it is known that there is no diffeomorphism from the circle to the unit interval. Thus (smooth) single parameter encodings cannot cover the entire circle, though the region of the circle left unparameterized can be made arbitrarily small. Depending on initial conditions, our technique found one of three different solutions. Some simulations resulted in a two–dimensional representation with the encodings lying on a circle in $\mathbf{R}^2$. This is a failure to reduce the dimensionality. The other solutions were both 1–D representations; one "wrapping" the unit interval around the circle, the other "splitting" the interval into two pieces. The initial architecture consisted of a single 8-unit encoding layer and two 8-unit decoding layers. $\eta$ was set to 0.01, $\Delta\lambda$ to 0.1, and the error threshold, $\epsilon$, to 0.001.

The helix problem is interesting because the data appears to be three–dimensional to PCA. NLDR consistently finds an invertible one–dimensional representation of the data. Figure 2

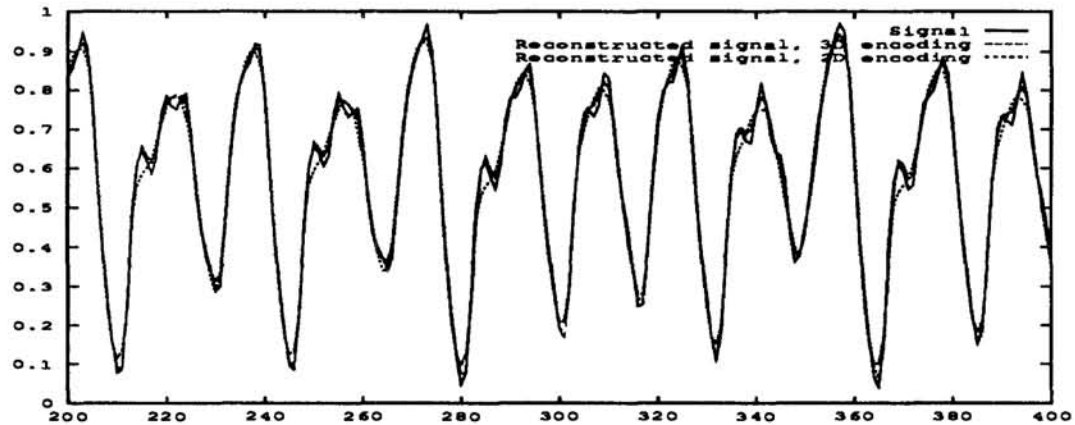

Figure 3: *Data from the Mackey–Glass delay–differential equation with $\tau = 17$, correlation dimension 2.1, and the reconstructed signal encoded in two and three dimensions.*

shows the original data, along with the network's output when the representation layer was stimulated with activation ranging from 0.1 to 0.9. The training data were mapped into the interval $0.213 - 0.778$ using a single (sigmoidal) representation unit. The initial architecture consisted of a single 10-unit encoding layer and two 10-unit decoding layers. $\eta$ was set to 0.01, $\Delta\lambda$ to 0.1, and the error threshold, $\epsilon$, to 0.001.

## 3.2   NLDR Applied to Time Series

The Mackey–Glass problem consists of estimation of the intrinsic dimensionality of a scalar signal. Classically, such time series data is embedded in a space of "high enough" dimension such that one expects the geometric invariants to be preserved. However, this may significantly overestimate the number of variables needed to describe the data. Two different series were examined; parameter settings for the Mackey–Glass equation were chosen such that the intrinsic dimensionality is 2.1 and 3.5. The data was embedded in a high dimensional space by the standard technique of recoding as vectors of lagged data. A 3 dimensional representation was found for the 2.1 dimensional data and a 4 dimensional representation was found for the 3.5 dimensional data. Figure 3 shows the original data and its reconstruction for the 2.1 dimensional data. Allowing higher reconstruction error resulted in a 3 dimensional representation for the 3.5 dimensional data, effectively smoothing the original signal (DeMers, 1992). Figure 4 shows the original data and its reconstruction for the 3.5 dimensional data. The initial architecture consisted of a two 10-unit encoding layers and two 10-unit decoding layers, and a 7-unit representation layer. The representation layer was connected directly to the output layer. $\eta$ was set to 0.01, $\Delta\lambda$ to 0.1, and the error threshold, $\epsilon$, to 0.001.

## 3.3   Faces

The face image data is much more challenging. The face data are $64 \times 64$ pixel, 8–bit grayscale images taken from (Cottrell & Metcalfe, 1991), each of which can be considered to be a point in a 4,096 dimensional "pixel space". The question addressed is whether NLDR can find low–dimensional representations of the data which are more useful than principal components. The data was preprocessed by reduction to the first 50 principal

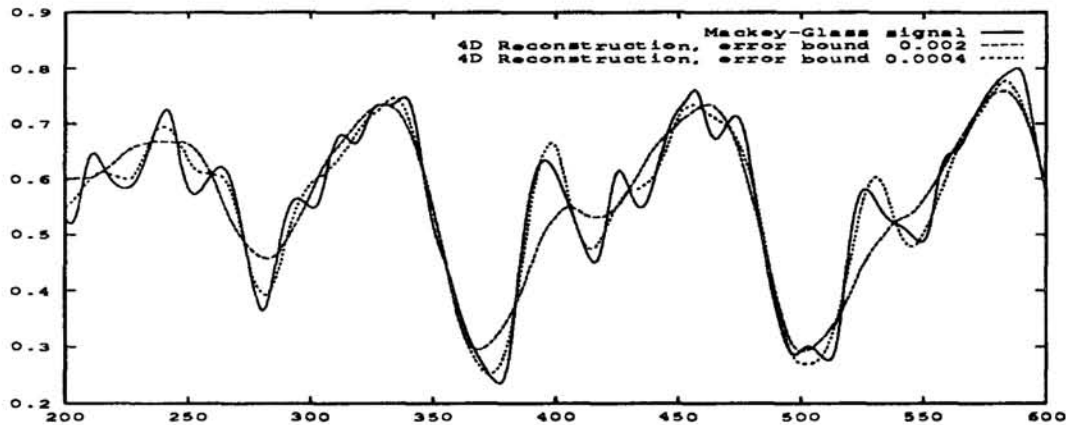

Figure 4: *Data from the Mackey–Glass delay–differential equation with* $\tau = 35$, *correlation dimension 3.5, and the reconstructed signal encoded in four dimensions with two different error thresholds.*

components[3] of the images. These reduced representations were then processed further by NLDR. The architecture consisted of a 30-unit encoding layer and a 30-unit decoding layer, and an initial representation layer of 20 units. There were direct connections from the representation layer to the output layer. $\eta$ was 0.05, $\Delta\lambda$ was 0.1 and $\epsilon$ was 0.001. NLDR found a five–dimensional representation. Figure 5 shows four of the 160 images after reduction to the first 50 principal components (used as training) and the same images after reconstruction from a five dimensional encoding. We are unable to determine whether the dimensions are meaningful; however, experiments with the decoder show that points inside the convex hull of the representations project to images which look like faces. Figure 6 shows the reconstructed images from a linear interpolation in "face space" between the two encodings which are furthest apart.

How useful are the representations obtained from a training set for identification and classification of other images of the same subjects? The 5–D representations were used to train a feedforward network to recognize the identity and gender of the subjects, as in (Cottrell & Metcalfe, 1991). 120 images were used in training and the remaining 40 used as a test set. The network correctly identified 98% of the training data subjects, and 95% on the test set. The network achieved 95% correct gender recognition on both the training and test sets. The misclassified subject is shown in Figure 7. An informal poll of visitors to the poster in Denver showed that about 2/3 of humans classify the subject as male and 1/3 as female.

Although NLDR resulted in five dimensional encodings of the face data, and thus super-ficially compresses the data to approximately 55 bits per image or 0.013 bits per pixel, there is no data compression. Both the decoder portion of the network and the eigenvectors used in the initial processing must also be stored. These amortize to about 6 bits per pixel, whereas the original images require only 1.1 bits per pixel under run–length encoding. In order to achieve data compression, a much larger data set must be obtained in order to find the underlying human face manifold.

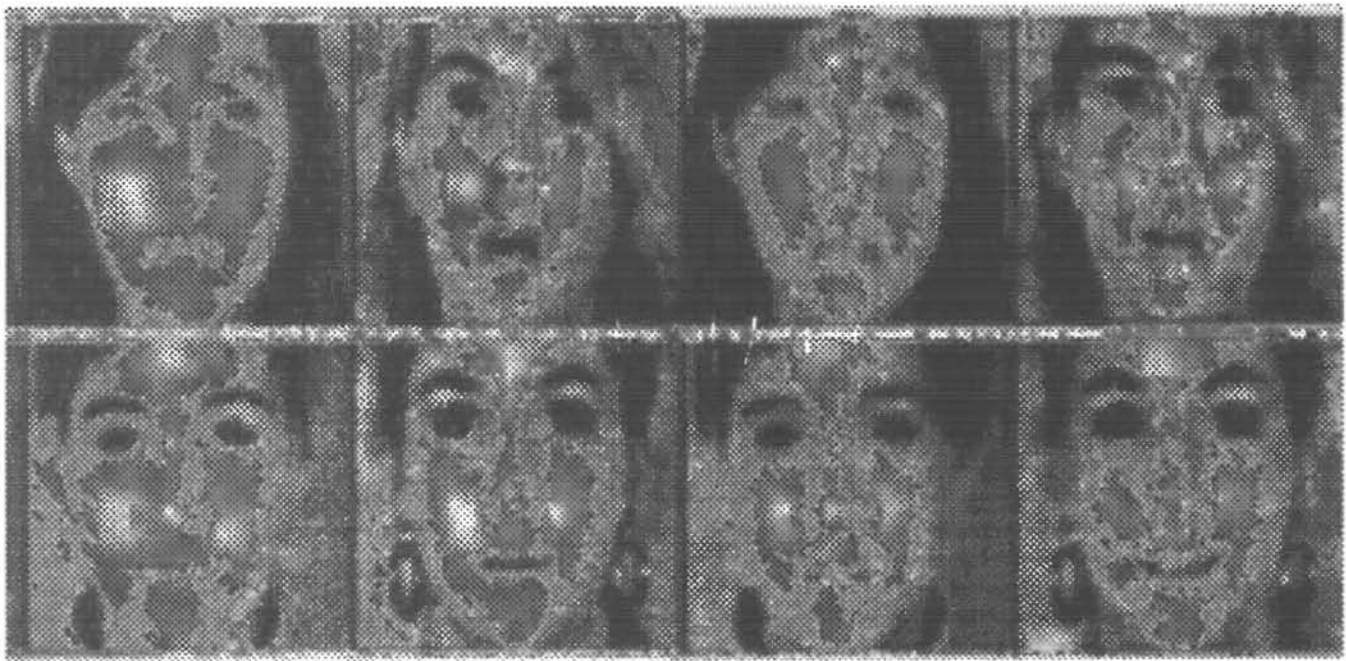

Figure 5: *Four of the original face images and their reconstruction after encoding as five dimensional data.*

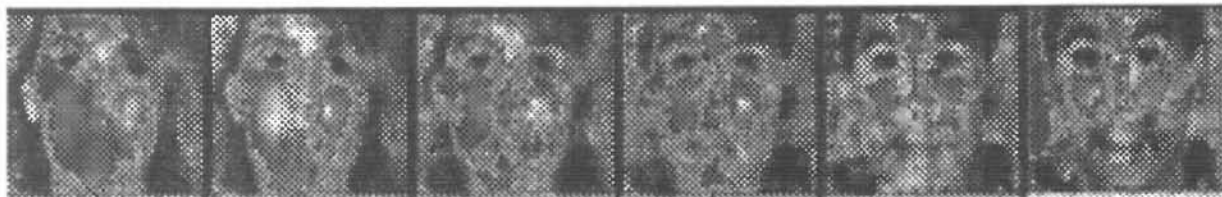

Figure 6: *The two images with 5-D encodings which are the furthest apart, and the reconstructions of four 5-D points equally spaced along the line joining them.*

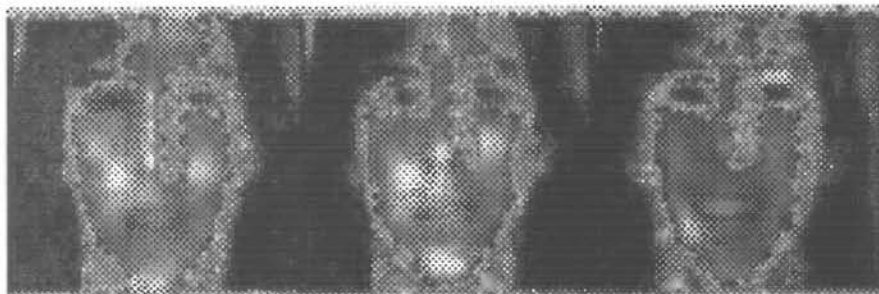

Figure 7: *"Pat"; the subject whose gender a feedforward network classified incorrectly.*

# 4   CONCLUSIONS

A method for automatically generating a non–linear encoder/decoder for high dimensional data has been presented. The number of representation units in the final network is an estimate of the intrinsic dimensionality of the data. The results are sensitive to the choice of error bound, though the precise relationship is as yet unknown. The size of the encoding and decoding hidden layers must be controlled to avoid over–fitting; any data set can be encoded into scalar values given enough resolution. Since we are using gradient search to solve a global non–linear optimization problem, there is no guarantee that this method will find the global optimum and avoid convergence to local minima. However, NLDR consistently constructed low dimensional encodings which were decodeable with low loss.

### Acknowledgements

We would like to thank Matthew Turk & Alex Pentland for making their *facerec* software available, which was used to extract the eigenvectors of the original face data. The first author was partially supported by Fellowships from the California Space Institute and the McDonnell–Pew Foundation.

## Footnotes

*email: demers@cs.ucsd.edu

[1]There is no reason to suppose that the encoding and decoding layers must be of the same size. In fact, it may be that two encoding or decoding layers will provide superior performance. For the helix example, the decoder had two hidden layers and linear connections from the representation to the output, while the encoder had a single layer. Kramer (1991) uses information theoretic measures for choosing the size of the encoding and decoding layers; however, only a fixed representation layer and equal encoding and decoding layers are used.

[2]Unbounded weights will allow the same amount of information to pass through the layer with arbitrarily small variance and using arbitrarily large weights. Therefore the weights in the network must be bounded. Weight vectors with magnitudes larger than 10 are renormalized after each epoch.

[3]50 was chosen by eyeballing a graph of the eigenvalues for the point at which they began to "flatten"; any value between about 40 and 80 would be reasonable.

### References

Pierre Baldi and Kurt Hornik (1988) "Neural Networks and Principal Component Analysis: Learning from Examples without Local Minima", *Neural Networks* 2, 53–58.

Garrison Cottrell and Paul Munro (1988) "Principal Components Analysis of Images via Backpropagation", in *Proc. SPIE* (Cambridge, MA).

Garrison Cottrell, Paul Munro, and David Zipser (1989) "Image Compression by Backpropagation: A Demonstration of Extensional Programming", In Sharkey, Noel (Ed.), *Models of Cognition: A review of Cognitive Science*, vol. 1.

Garrison Cottrell and Janet Metcalfe (1991) "EMPATH — Face, Emotion and Gender Recognition using Holons" in Lippmann, R., Moody, J. & Touretzky, D., (eds), *Advances in Neural Information Processing Systems* 3.

David DeMers (1992) "Dimensionality Reduction for Non–Linear Time Series", *Neural and Stochastic Methods in Image and Signal Processing* (SPIE 1766).

Mark Kramer (1991) "Nonlinear Principal Component Analysis Using Autoassociative Neural Networks", *AIChE Journal* 37:233-243.

Erkki Oja (1991) "Data Compression, Feature Extraction, and Autoassociation in Feedforward Neural Networks" in Kohonen, T., Simula, O. and Kangas, J., eds, *Artificial Neural Networks*, 737-745.

Shiro Usui, Shigeki Nakauchi, and Masae Nakano (1991) "Internal Color Representation Acquired by a Five–Layer Neural Network", in Kohonen, T., Simula, O. and Kangas, J., eds, *Artificial Neural Networks*, 867-872.
